# $t$-divergence Based Approximate Inference

**Nan Ding**[2]**, S.V. N. Vishwanathan**[1,2]**, Yuan Qi**[2,1]
Departments of [1]Statistics and [2]Computer Science
Purdue University
ding10@purdue.edu, vishy@stat.purdue.edu, alanqi@cs.purdue.edu

## Abstract

Approximate inference is an important technique for dealing with large, intractable graphical models based on the exponential family of distributions. We extend the idea of approximate inference to the $t$-exponential family by defining a new $t$-divergence. This divergence measure is obtained via convex duality between the log-partition function of the $t$-exponential family and a new $t$-entropy. We illustrate our approach on the Bayes Point Machine with a Student's $t$-prior.

## 1 Introduction

The exponential family of distributions is ubiquitous in statistical machine learning. One prominent application is their use in modeling conditional independence between random variables via a graphical model. However, when the number or random variables is large, and the underlying graph structure is complex, a number of computational issues need to be tackled in order to make inference feasible. Therefore, a number of approximate techniques have been brought to bear on the problem. Two prominent approximate inference techniques include the Monte Carlo Markov Chain (MCMC) method [1], and the deterministic method [2, 3].

Deterministic methods are gaining significant research traction, mostly because of their high efficiency and practical success in many applications. Essentially, these methods are premised on the search for a proxy in an analytically solvable distribution family that approximates the true underlying distribution. To measure the closeness between the true and the approximate distributions, the relative entropy between these two distributions is used. When working with the exponential family, one uses the Shannon-Boltzmann-Gibbs (SBG) entropy in which case the relative entropy is the well known Kullback-Leibler (KL) divergence [2]. Numerous well-known algorithms in exponential family, such as the mean field method [2, 4] and the expectation propagation [3, 5], are based on this criterion.

The thin-tailed nature of the exponential family makes it unsuitable for designing algorithms which are potentially robust against certain kinds of noisy data. Notable work including [6, 7] utilizes mixture/split exponential family based approximate model to improve the robustness. Meanwhile, effort has also been devoted to develop alternate, generalized distribution families in statistics [e.g. 8, 9], statistical physics [e.g. 10, 11], and most recently in machine learning [e.g. 12]. Of particular interest to us is the $t$-exponential family[1], which was first proposed by Tsallis and co-workers [10, 13, 14]. It is a special case of the more general $\phi$-exponential family of Naudts [11, 15–17]. Related work in [18] has applied the $t$-exponential family to generalize logistic regression and obtain an algorithm that is robust against certain types of label noise.

In this paper, we attempt to generalize deterministic approximate inference by using the $t$-exponential family. In other words, the approximate distribution used is from the $t$-exponential family. To obtain the corresponding divergence measure as in the exponential family, we exploit the

convex duality between the log-partition function of the $t$-exponential family and a new $t$-entropy[2] to define the $t$-divergence. To illustrate the usage of the above procedure, we use it for approximate inference in the Bayes Point Machine (BPM) [3] but with a Student's $t$-prior.

The rest of the paper is organized as follows. Section 2 consists of a brief review of the $t$-exponential family. In Section 3 a new $t$-entropy is defined as the convex dual of the log-partition function of the $t$-exponential family. In Section 4, the $t$-divergence is derived and is used for approximate inference in Section 5. Section 6 illustrates the inference approach by applying it to the Bayes Point Machine with a Student's $t$-prior, and we conclude the paper with a discussion in Section 7.

## 2   The $t$-exponential Family and Related Entropies

The $t$-exponential family was first proposed by Tsallis and co-workers [10, 13, 14]. It is defined as

$$p(x; \theta) := \exp_t \left( \langle \Phi(x), \theta \rangle - g_t(\theta) \right), \text{ where} \tag{1}$$

$$\exp_t(x) := \begin{cases} \exp(x) & \text{if } t = 1 \\ [1 + (1-t)x]_+^{\frac{1}{1-t}} & \text{otherwise.} \end{cases} \tag{2}$$

The inverse of the $\exp_t$ function is called $\log_t$. Note that the log-partition function, $g_t(\theta)$, in (1) preserves convexity and satisfies

$$\nabla_\theta g_t(\theta) = \mathbb{E}_q \left[ \Phi(x) \right]. \tag{3}$$

Here $q(x)$ is called the escort distribution of $p(x)$, and is defined as

$$q(x) := \frac{p(x)^t}{\int p(x)^t dx}. \tag{4}$$

See the supplementary material for a proof of convexity of $g_t(\theta)$ based on material from [17], and a detailed review of the $t$-exponential family of distributions.

There are various generalizations of the Shannon-Boltzmann-Gibbs (SBG) entropy which are proposed in statistical physics, and paired with the $t$-exponential family of distributions. Perhaps the most well-known among them is the Tsallis entropy [10]:

$$H_{\text{tsallis}}(p) := - \int p(x)^t \log_t p(x) dx. \tag{5}$$

Naudts [11, 15, 16, 17] proposed a more general framework, wherein the familiar $\exp$ and $\log$ functions are generalized to $\exp_\phi$ and $\log_\phi$ functions which are defined via a function $\phi$. These generalized functions are used to define a family of distributions, and corresponding to this family an entropy like measure called the information content $I_\phi(p)$ as well as its divergence measure are defined. The information content is the dual of a function $F(\theta)$, where

$$\nabla_\theta F(\theta) = \mathbb{E}_p \left[ \Phi(x) \right]. \tag{6}$$

Setting $\phi(p) = p^t$ in the Naudts framework recovers the $t$-exponential family defined in (1). Interestingly when $\phi(p) = \frac{1}{t}p^{2-t}$, the information content $I_\phi$ is exactly the Tsallis entropy (5).

One another well-known non-SBG entropy is the Rényi entropy [19]. The Rényi $\alpha$-entropy (when $\alpha \neq 1$) of the probability distribution $p(x)$ is defined as:

$$H_\alpha(p) = \frac{1}{1-\alpha} \log \left( \int p(x)^\alpha dx \right). \tag{7}$$

Besides these entropies proposed in statistical physics, it is also worth noting efforts that work with generalized linear models or utilize different divergence measures, such as [5, 8, 20, 21].

It is well known that the negative SBG entropy is the Fenchel dual of the log-partition function of an exponential family distribution. This fact is crucially used in variational inference [2]. Although all

of the above generalized entropies are useful in their own way, none of them satisfy this important property for the $t$-exponential family. In the following sections we attempt to find an entropy which satisfies this property, and outline the principles of approximate inference using the $t$-exponential family. Note that although our main focus is the $t$-exponential family, we believe that our results can also be extended to the more general $\phi$-exponential family of Naudts [15, 17].

## 3   Convex Duality and the $t$-Entropy

**Definition 1 (Inspired by Wainwright and Jordan [2])** *The $t$-entropy of a distribution $p(x; \theta)$ is defined as*

$$H_t(p(x; \theta)) := -\int q(x; \theta) \log_t p(x; \theta)\, dx = -\mathbb{E}_q\left[\log_t p(x; \theta)\right].\tag{8}$$

where $q(x; \theta)$ is the escort distribution of $p(x; \theta)$. It is straightforward to verify that the $t$-entropy is non-negative. Furthermore, the following theorem establishes the duality between $-H_t$ and $g_t$. The proof is provided in the supplementary material. This extends Theorem 3.4 of [2] to the $t$-entropy.

**Theorem 2** *For any $\mu$, define $\theta(\mu)$ (if exists) to be the parameter of the $t$-exponential family s.t.*

$$\mu = \mathbb{E}_{q(x; \theta(\mu))}\left[\Phi(x)\right] = \int \Phi(x) q(x; \theta(\mu))\, dx.\tag{9}$$

$$Then \quad g_t^*(\mu) = \begin{cases} -H_t(p(x; \theta(\mu))) \text{ if } \theta(\mu) \text{ exists} \\ +\infty \text{ otherwise} . \end{cases}\tag{10}$$

*where $g_t^*(\mu)$ denotes the Fenchel dual of $g_t(\theta)$. By duality it also follows that*

$$g_t(\theta) = \sup_{\mu} \left\{\langle \mu, \theta \rangle - g_t^*(\mu)\right\}.\tag{11}$$

From Theorem 2, it is obvious that $H_t(\mu)$ is a concave function. Below, we derive the $t$-entropy function corresponding to two commonly used distributions. See Figure 1 for a graphical illustration.

**Example 1 ($t$-entropy of Bernoulli distribution)** *Assume the Bernoulli distribution is $Bern(p)$ with parameter $p$. The $t$-entropy is*

$$H_t(p) = \frac{-p^t \log_t p - (1-p)^t \log_t(1-p)}{p^t + (1-p)^t} = \frac{1 - (p^t + (1-p)^t)^{-1}}{t-1}\tag{12}$$

**Example 2 ($t$-entropy of Student's $t$-distribution)** *Assume that a $k$-dim Student's $t$-distribution $p(\mathbf{x}; \boldsymbol{\mu}, \boldsymbol{\Sigma}, v)$ is given by (54), then the $t$-entropy of $p(\mathbf{x}; \boldsymbol{\mu}, \boldsymbol{\Sigma}, v)$ is given by*

$$H_t(p(\mathbf{x}))) = -\frac{\Psi}{1-t}\left(1 + v^{-1}\right) + \frac{1}{1-t}\tag{13}$$

*where $\mathbf{K} = (v\,\boldsymbol{\Sigma})^{-1}$, $v = \frac{2}{t-1} - k$, and $\Psi = \left(\frac{\Gamma((v+k)/2)}{(\pi v)^{k/2}\Gamma(v/2)|\boldsymbol{\Sigma}|^{1/2}}\right)^{-2/(v+k)}$.*

### 3.1   Relation with the Tsallis Entropy

Using (4), (5), and (8), the relation between the $t$-entropy and Tsallis entropy is obvious. Basically, the $t$-entropy is a normalized version of the Tsallis entropy,

$$H_t(p) = -\frac{1}{\int p(x)^t dx}\int p(x)^t \log_t p(x) dx = \frac{1}{\int p(x)^t dx} H_{\text{tsallis}}(p).\tag{14}$$

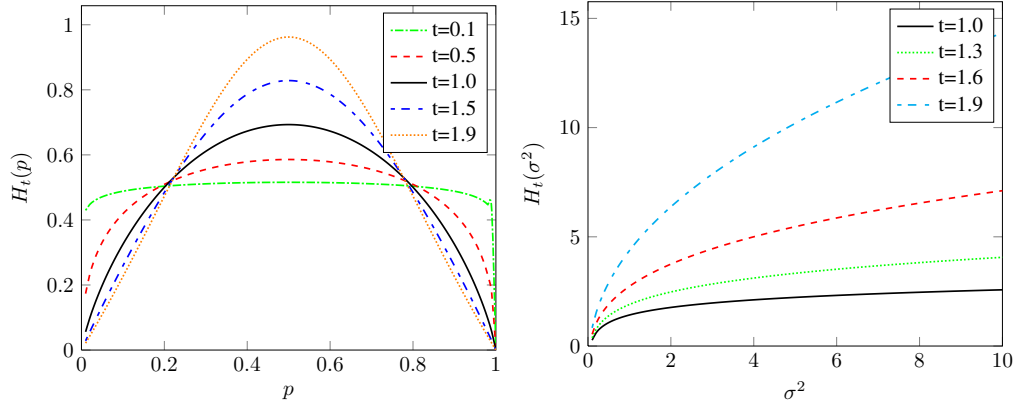

Figure 1: $t$-entropy corresponding to two well known probability distributions. Left: the Bernoulli distribution $Bern(x; p)$; Right: the Student's $t$-distribution $St(x; 0, \sigma^2, v)$, where $v = 2/(t-1) - 1$. One can recover the SBG entropy by setting $t = 1.0$.

### 3.2 Relation with the Rényi Entropy

We can equivalently rewrite the Rényi Entropy as:

$$H_\alpha(p) = \frac{1}{1-\alpha} \log \left( \int p(x)^\alpha dx \right) = -\log \left( \int p(x)^\alpha dx \right)^{-1/(1-\alpha)}. \tag{15}$$

The $t$-entropy of $p(x)$ (when $t \neq 1$) is equal to

$$H_t(p) = -\frac{\int p(x)^t \log_t p(x) dx}{\int p(x)^t dx} = -\log_t \left( \int p(x)^t dx \right)^{-1/(1-t)}. \tag{16}$$

Therefore, when $\alpha = t$,

$$H_t(p) = -\log_t(\exp(-H_\alpha(p))) \tag{17}$$

When $t$ and $\alpha \to 1$, both entropies go to the SBG entropy.

## 4 The $t$-divergence

Recall that the Bregman divergence defined by a convex function $-H$ between $p$ and $\tilde{p}$ is [22]:

$$D(p\|\tilde{p}) = -H(p) + H(\tilde{p}) + \int \frac{dH(\tilde{p})}{d\tilde{p}}(\tilde{p}(x) - p(x)) dx. \tag{18}$$

For the SBG entropy, it is easy to verify that the Bregman divergence leads to the relative SBG-entropy (also widely known as the Kullback-Leibler (KL) divergence). Analogously, one can define the $t$-divergence[3] as the Bregman divergence or relative entropy based on the $t$-entropy.

**Definition 3** *The $t$-divergence, which is the relative $t$-entropy between two distribution $p(x)$ and $\tilde{p}(x)$, is defined as,*

$$D_t(p\|\tilde{p}) = \int q(x) \log_t p(x) - q(x) \log_t \tilde{p}(x) dx. \tag{19}$$

The following theorem states the relationship between the relative $t$-entropy and the Bregman divergence. The proof is provided in the supplementary material.

**Theorem 4** *The $t$-divergence is the Bregman divergence defined on the negative $t$-entropy $-H_t(p)$.*

The $t$-divergence plays a central role in the variational inference that will be derived shortly. It also preserves the following properties:

- $D_t(p\|\tilde{\mathrm{p}}) \geq 0, \forall p, \tilde{\mathrm{p}}$. The equality holds only for $p = \tilde{\mathrm{p}}$.
- $D_t(p\|\tilde{\mathrm{p}}) \neq D_t(\tilde{\mathrm{p}}\|p)$.

**Example 3 (Relative $t$-entropy between Bernoulli distributions)** *Assume that two Bernoulli distributions $Bern(p_1)$ and $Bern(p_2)$, then the relative t-entropy $D_t(p_1\|p_2)$ between these two distributions is:*

$$D_t(p_1\|p_2) = \frac{p_1^t \log_t p_1 + (1-p_1)^t \log_t(1-p_1) - p_1^t \log_t p_2 - (1-p_1)^t \log_t(1-p_2)}{p_1^t + (1-p_1)^t} \tag{20}$$

$$= \frac{1 - p_1^t p_2^{1-t} - (1-p_1)^t (1-p_2)^{1-t}}{(1-t)(p_1^t + (1-p_1)^t)} \tag{21}$$

**Example 4 (Relative $t$-entropy between Student's $t$-distributions)** *Assume that two Student's $t$-distributions $p_1(\mathbf{x}; \boldsymbol{\mu}_1, \boldsymbol{\Sigma}_1, v)$ and $p_2(\mathbf{x}; \boldsymbol{\mu}_2, \boldsymbol{\Sigma}_2, v)$ are given, then the relative t-entropy $D_t(p_1\|p_2)$ between these two distributions is:*

$$D_t(p_1\|p_2) = \int q_1(x) \log_t p_1(x) - q_1(x) \log_t p_2(x) dx$$

$$= \frac{\Psi_1}{1-t}\left(1 + v^{-1}\right) + \frac{2\Psi_2}{1-t}\boldsymbol{\mu}_1^\top \mathbf{K}_2\,\boldsymbol{\mu}_2 \tag{22}$$

$$- \frac{\Psi_2}{1-t}Tr\left(\mathbf{K}_2^\top \boldsymbol{\Sigma}_1\right) - \frac{\Psi_2}{1-t}\boldsymbol{\mu}_1^\top \mathbf{K}_2\,\boldsymbol{\mu}_1 - \frac{\Psi_2}{1-t}\left(\boldsymbol{\mu}_2^\top \mathbf{K}_2\,\boldsymbol{\mu}_2 + 1\right) \tag{23}$$

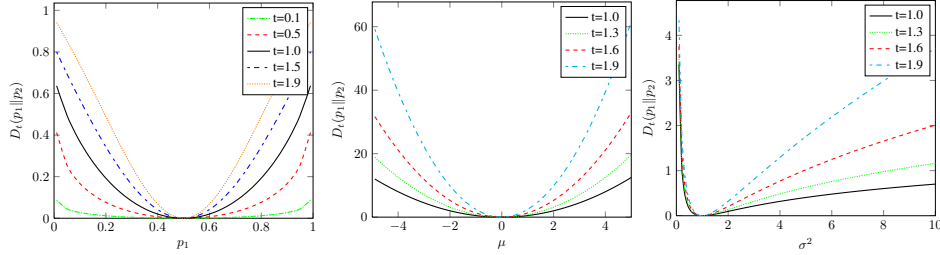

Figure 2: The $t$-divergence between: Left: $Bern(p_1)$ and $Bern(p_2 = 0.5)$; Middle: $St(x; \mu, 1, v)$ and $St(x; 0, 1, v)$; Right: $St(x; 0, \sigma^2, v)$ and $St(x; 0, 1, v)$, where $v = 2/(t-1) - 1$.

# 5  Approximate Inference in the $t$-Exponential Family

In essence, the deterministic approximate inference finds an approximate distribution from an analytically tractable distribution family which minimizes the relative entropy (*e.g.* KL-divergence in exponential family) with the true distribution. Since the relative entropy is not symmetric, the results of minimizing $D(p\|\tilde{\mathrm{p}})$ and $D(\tilde{\mathrm{p}}\|p)$ are different. In the main body of the paper we describe methods which minimize $D(p\|\tilde{\mathrm{p}})$ where $\tilde{\mathrm{p}}$ comes from the $t$-exponential family. Algorithms which minimize $D(\tilde{\mathrm{p}}\|p)$ are described in the supplementary material.

Given an arbitrary probability distribution $p(x)$, in order to obtain a good approximation $\tilde{\mathrm{p}}(x; \theta)$ in the $t$-exponential family, we minimize the relative $t$-relative entropy (19)

$$\tilde{\mathrm{p}} = \underset{\tilde{\mathrm{p}}}{\mathrm{argmin}}\, D_t(p\|\tilde{\mathrm{p}}) = \int q(x) \log_t p(x) - q(x) \log_t \tilde{\mathrm{p}}(x; \theta) dx. \tag{24}$$

Here $q(x) = \frac{1}{Z}p(x)^t$ denotes the escort of the original distribution $p(x)$. Since

$$\tilde{\mathrm{p}}(x; \theta) = \exp_t(\langle \Phi(x), \theta \rangle - g_t(\theta)), \tag{25}$$

using the fact that $\nabla_\theta g_t(\theta) = \mathbb{E}_{\tilde{q}}[\Phi(x)]$, one can take the derivative of (24) with respect to $\theta$:

$$\mathbb{E}_q[\Phi(x)] = \mathbb{E}_{\tilde{q}}[\Phi(x)]. \tag{26}$$

In other words, the approximate distribution can be obtained by matching the *escort* expectation of $\Phi(x)$ between the two distributions.

The escort expectation matching in (26) is reminiscent of the moment matching in the Power-EP [5] or the Fractional BP [23] algorithm, where the approximate distribution is obtained by

$$\mathbb{E}_{\tilde{p}}[\Phi(x)] = \mathbb{E}_{p^\alpha \, \tilde{p}^{1-\alpha} \, /Z}[\Phi(x)]. \tag{27}$$

The main reason for using the $t$-divergence, however, is not to address the computational or convergence issues as is done in the case of power EP/fractional BP. In contrast, we use the generalized exponential family ($t$-exponential family) to build our approximate models. In this context, the $t$-divergence plays the same role as KL divergence in the exponential family.

To illustrate our ideas on a non-trivial problem, we apply escort expectation matching to the Bayes Point Machine (BPM) [3] with a Student's $t$-distribution prior.

# 6   Bayes Point Machine with Student's $t$-Prior

Let $\mathcal{D} = \{(\mathbf{x}_1, y_1), \ldots, (\mathbf{x}_n, y_n)\}$ be the training data. Consider a linear model parametrized by the $k$-dim weight vector $\mathbf{w}$. For each training data point $(\mathbf{x}_i, y_i)$, the conditional distribution of the label $y_i$ given $\mathbf{x}_i$ and $\mathbf{w}$ is modeled as [3]:

$$t_i(\mathbf{w}) = p(y_i \mid \mathbf{x}_i, \mathbf{w}) = \epsilon + (1 - 2\epsilon)\Theta(y_i \langle \mathbf{w}, \mathbf{x}_i \rangle), \tag{28}$$

where $\Theta(z)$ is the step function: $\Theta(z) = 1$ if $z > 0$ and $= 0$ otherwise. By making a standard i.i.d. assumption about the data, the posterior distribution can be written as

$$p(\mathbf{w} \mid \mathcal{D}) \propto p_0(\mathbf{w}) \prod_i t_i(\mathbf{w}), \tag{29}$$

where $p_0(\mathbf{w})$ denotes a prior distribution. Instead of using multivariate Gaussian distribution as a prior as was done by Minka [3], we will use a Student's $t$-prior, because we want to build robust models:

$$p_0(\mathbf{w}) = St(\mathbf{w}; \mathbf{0}, \mathbf{I}, v). \tag{30}$$

As it turns out, the posterior $p(\mathbf{w} \mid \mathcal{D})$ is infeasible to obtain in practice. Therefore we will find a multivariate Student's $t$-distribution to approximate the true posterior.

$$p(\mathbf{w} \mid \mathcal{D}) \simeq \tilde{p}(\mathbf{w}) = St(\mathbf{w}; \tilde{\boldsymbol{\mu}}, \tilde{\boldsymbol{\Sigma}}, v). \tag{31}$$

In order to obtain such a distribution, we implement the Bayesian online learning method [24], which is also known as Assumed Density Filter [25]. The extension to the expectation propagation is similar to [3] and omitted due to space limitation. The main idea is to process data points one by one and update the posterior by using *escort* moment matching. Assume the approximate distribution after processing $(\mathbf{x}_1, y_1), \ldots, (\mathbf{x}_{i-1}, y_{i-1})$ to be $\tilde{p}_{i-1}(\mathbf{w})$ and define

$$\tilde{p}_0(\mathbf{w}) = p_0(\mathbf{w}) \tag{32}$$
$$p_i(\mathbf{w}) \propto \tilde{p}_{i-1}(\mathbf{w})t_i(\mathbf{w}) \tag{33}$$

Then the approximate posterior $\tilde{p}_i(\mathbf{w})$ is updated as

$$\tilde{p}_i(\mathbf{w}) = St(\mathbf{w}; \boldsymbol{\mu}^{(i)}, \boldsymbol{\Sigma}^{(i)}, v) = \underset{\boldsymbol{\mu}, \boldsymbol{\Sigma}}{\operatorname{argmin}} \, D_t(p_i(\mathbf{w}) \| St(\mathbf{w}; \boldsymbol{\mu}, \boldsymbol{\Sigma}, v)). \tag{34}$$

Because $\tilde{p}_i(\mathbf{w})$ is a $k$-dim Student's $t$-distribution with degree of freedom $v$, for which $\Phi(\mathbf{w}) = [\mathbf{w}, \mathbf{w}\mathbf{w}^\top]$ and $t = 1 + 2/(v + k)$ (see example 5 in Appendix A), it turns out that we only need

$$\int q_i(\mathbf{w}) \, \mathbf{w} \, d\mathbf{w} = \int \tilde{q}_i(\mathbf{w}) \, \mathbf{w} \, d\mathbf{w}, \text{ and} \tag{35}$$

$$\int q_i(\mathbf{w}) \, \mathbf{w} \mathbf{w}^\top \, d\mathbf{w} = \int \tilde{q}_i(\mathbf{w}) \, \mathbf{w} \mathbf{w}^\top \, d\mathbf{w}. \tag{36}$$

Here $\tilde{q}_i(\mathbf{w}) \propto \tilde{p}_i(\mathbf{w})^t$, $q_i(\mathbf{w}) \propto \tilde{p}_{i-1}(\mathbf{w})^t \tilde{t}_i(\mathbf{w})$ and

$$\tilde{t}_i(\mathbf{w}) = t_i(\mathbf{w})^t = \epsilon^t + \left((1-\epsilon)^t - \epsilon^t\right) \Theta(y_i \langle \mathbf{w}, \mathbf{x}_i \rangle). \tag{37}$$

Denote $\tilde{p}_{i-1}(\mathbf{w}) = St(\mathbf{w}; \boldsymbol{\mu}^{(i-1)}, \boldsymbol{\Sigma}^{(i-1)}, v)$, $\tilde{q}_{i-1}(\mathbf{w}) = St(\mathbf{w}; \boldsymbol{\mu}^{(i-1)}, v\,\boldsymbol{\Sigma}^{(i-1)}/(v+2), v+2)$ (also see example 5), and we make use of the following relations:

$$Z_1 = \int \tilde{p}_{i-1}(\mathbf{w}) \tilde{t}_i(\mathbf{w}) d\mathbf{w} \tag{38}$$

$$= \epsilon^t + \left((1-\epsilon)^t - \epsilon^t\right) \int_{-\infty}^{z} St(x; 0, 1, v) dx \tag{39}$$

$$Z_2 = \int \tilde{q}_{i-1}(\mathbf{w}) \tilde{t}_i(\mathbf{w}) d\mathbf{w} \tag{40}$$

$$= \epsilon^t + \left((1-\epsilon)^t - \epsilon^t\right) \int_{-\infty}^{z} St(x; 0, v/(v+2), v+2) dx \tag{41}$$

$$\mathbf{g} = \frac{1}{Z_2} \nabla_{\boldsymbol{\mu}} Z_1 = y_i \alpha \, \mathbf{x}_i \tag{42}$$

$$\mathbf{G} = \frac{1}{Z_2} \nabla_{\boldsymbol{\Sigma}} Z_1 = -\frac{1}{2} \frac{y_i \alpha \langle \mathbf{x}_i, \boldsymbol{\mu}^{(i-1)} \rangle}{\mathbf{x}_i^\top \boldsymbol{\Sigma}^{(i-1)} \mathbf{x}_i} \mathbf{x}_i \mathbf{x}_i^\top \tag{43}$$

where,

$$\alpha = \frac{\left((1-\epsilon)^t - \epsilon^t\right) St(z; 0, 1, v)}{Z_2 \sqrt{\mathbf{x}_i^\top \boldsymbol{\Sigma}^{(i-1)} \mathbf{x}_i}} \quad \text{and} \quad z = \frac{y_i \langle \mathbf{x}_i, \boldsymbol{\mu}^{(i-1)} \rangle}{\sqrt{\mathbf{x}_i^\top \boldsymbol{\Sigma}^{(i-1)} \mathbf{x}_i}}.$$

Equations (39) and (41) are analogous to Eq. (5.17) in [3]. By assuming that a regularity condition[4] holds, $\int$ and $\nabla$ can be interchanged in $\nabla Z_1$ of (42) and (43). Combining with (38) and (40), we obtain the escort expectations of $p_i(\mathbf{w})$ from $Z_1$ and $Z_2$ (similar to Eq. (5.12) and (5.13) in [3]),

$$\mathbb{E}_q[\mathbf{w}] = \frac{1}{Z_2} \int \tilde{q}_{i-1}(\mathbf{w}) \tilde{t}_i(\mathbf{w}) \mathbf{w} \, d\mathbf{w} = \boldsymbol{\mu}^{(i-1)} + \boldsymbol{\Sigma}^{(i-1)} \mathbf{g} \tag{44}$$

$$\mathbb{E}_q[\mathbf{w} \mathbf{w}^\top] - \mathbb{E}_q[\mathbf{w}] \mathbb{E}_q[\mathbf{w}]^\top = \frac{1}{Z_2} \int \tilde{q}_{i-1}(\mathbf{w}) \tilde{t}_i(\mathbf{w}) \mathbf{w} \mathbf{w}^\top \, d\mathbf{w} - \mathbb{E}_q[\mathbf{w}] \mathbb{E}_q[\mathbf{w}]^\top$$

$$= r\, \boldsymbol{\Sigma}^{(i-1)} - \boldsymbol{\Sigma}^{(i-1)} \left(\mathbf{g} \mathbf{g}^\top - 2\mathbf{G}\right) \boldsymbol{\Sigma}^{(i-1)} \tag{45}$$

where $r = Z_1/Z_2$ and $\mathbb{E}_q[\cdot]$ means the expectation with respect to $q_i(\mathbf{w})$.

Since the mean and variance of the escort of $\tilde{p}_i(\mathbf{w})$ is $\boldsymbol{\mu}^{(i)}$ and $\boldsymbol{\Sigma}^{(i)}$ (again see example 5), after combining with (42) and (43),

$$\boldsymbol{\mu}^{(i)} = \mathbb{E}_q[\mathbf{w}] = \boldsymbol{\mu}^{(i-1)} + \alpha y_i \boldsymbol{\Sigma}^{(i-1)} \mathbf{x}_i \tag{46}$$

$$\boldsymbol{\Sigma}^{(i)} = \mathbb{E}_q[\mathbf{w} \mathbf{w}^\top] - \mathbb{E}_q[\mathbf{w}] \mathbb{E}_q[\mathbf{w}]^\top = r\, \boldsymbol{\Sigma}^{(i-1)} - \left(\boldsymbol{\Sigma}^{(i-1)} \mathbf{x}_i\right) \left(\frac{\alpha y_i \langle \mathbf{x}_i, \boldsymbol{\mu}^{(i)} \rangle}{\mathbf{x}_i^\top \boldsymbol{\Sigma}^{(i-1)} \mathbf{x}_i}\right) \left(\boldsymbol{\Sigma}^{(i-1)} \mathbf{x}_i\right)^\top. \tag{47}$$

## 6.1 Results

In the above Bayesian online learning algorithm, everytime a new data $\mathbf{x}_n$ coming in, $p(\boldsymbol{\theta} \mid \mathbf{x}_1, \ldots, \mathbf{x}_{n-1})$ is used as a prior, and the posterior is computed by incorporating the likelihood $p(\mathbf{x}_n \mid \boldsymbol{\theta})$. The Student's $t$-distribution is a more *conservative* or *non-subjective* prior than the Gaussian distribution because its heavy-tailed nature. More specifically, it means that the Student's $t$-based BPM can be more strongly influenced by the newly coming in points.

In many binary classfication problems, it is assumed that the underlying classfication hyperplane is always fixed. However, in some real situations, this assumption might not hold. Especially, in

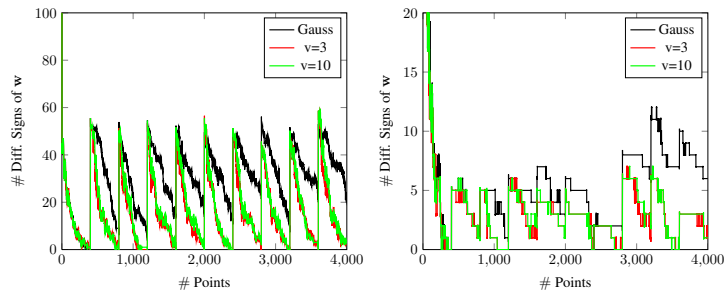

Figure 3: The number of wrong signs between $\mathbf{w}$. Left: case I; Right: case II

Table 1: The classification error of all the data points

|        | Gauss | v=3   | v=10  |
|--------|-------|-------|-------|
| Case I | 0.337 | **0.242** | 0.254 |
| Case II| 0.150 | 0.130 | **0.128** |

an online learning problem, the data sequence coming in is time dependent. It is possible that the underlying classifier is also time dependent. For a senario like this, we require our learning machine is able to self-adjust during the time given the data.

In our experiment, we build a synthetic online dataset which mimics the above senario, that is the underlying classification hyperplane is changed during a certain time interval. Our sequence of data is composed of 4000 data points randomly generated by a 100 dimension isotropic Gaussian distribution $\mathcal{N}(\mathbf{0}, \mathbf{I})$. The sequence can be partitioned into 10 sub-sequences of length 400. During each sub-sequence $s$, there is a base weight vector $\mathbf{w}_{(s)}^b \in \{-1, +1\}^{100}$. Each point $\mathbf{x}_{(i)}$ of the subsequence is labeled as $y_{(i)} = \text{sign}\left(\mathbf{w}_{(i)}^\top \mathbf{x}_{(i)}\right)$ where $\mathbf{w}_{(i)} = \mathbf{w}_{(s)}^b + \mathbf{n}$ and $\mathbf{n}$ is a random noise from $[-0.1, +0.1]^{100}$. The base weight vector $\mathbf{w}_{(s)}^b$ can be (I) totally randomly generated, or (II) generated based on the base weight vector $\mathbf{w}_{(s-1)}^b$ in the following way:

$$w_{(s)j}^b = \begin{cases} \text{Rand}\{-1, +1\} & j \in [400s - 399, 400s] \\ w_{(s-1)j}^b & \text{otherwise.} \end{cases} \tag{48}$$

Namely, only 10% of the base weight vector is changed based upon the previous base weight vector. We compare the Bayes Point Machine with Student's $t$-prior (with $v = 3$ and $v = 10$) with the Gaussian prior. For both method, $\epsilon = 0.01$. We report (1) for each point the number of different signs between the base weight vector and the mean of the posterior (2) the error rate of all the points.

According to the Fig. 3 and Table. 1, we find that the Bayes Point Machine with the Student's-$t$ prior adjusts itself significantly faster than the Gaussian prior and it also ends up with a better classification results. We believe that is mostly resulted from its heavy-tailness.

## 7 Discussion

In this paper, we investigated the convex duality of the log-partition function of the $t$-exponential family, and defined a new $t$-entropy. By using the $t$-divergence as a divergence measure, we proposed approximate inference on the $t$-exponential family by matching the expectation of the escort distributions. The results in this paper can be extended to the more generalized $\phi$-exponential family by Naudts [15].

The $t$-divergence based approximate inference is only applied in a toy example. The focus of our future work is on utilizing this approach in various graphical models. Especially, it is important to investigate a new family of graphical models based on heavy-tailed distributions for applications involving noisy data.

## Footnotes

[1]Sometimes, also called the $q$-exponential family or the Tsallis distribution.

[2]Although closely related, our $t$-entropy definition is different from either the Tsallis entropy [10] or the information content in [17]. Nevertheless, it can be regarded as an example of the generalized framework of the entropy proposed in [8].

[3]Note that the $t$-divergence is not a special case of the divergence measure of Naudts [17] because the entropies are defined differently the derivations are fairly similar in spirit.

[4]This is a fairly standard technical requirement which is often proved using the Dominated Convergence Theorem (see *e.g.* Section 9.2 of Rosenthal [26]).

# References

[1] W. R. Gilks, S. Richardson, and D. J. Spiegelhalter. *Markov Chain Monte Carlo in Practice*. Chapman & Hall, 1995.

[2] M. J. Wainwright and M. I. Jordan. Graphical models, exponential families, and variational inference. *Foundations and Trends in Machine Learning*, 1(1 − 2):1 − 305, 2008.

[3] T. Minka. *Expectation Propagation for approximative Bayesian inference*. PhD thesis, MIT Media Labs, Cambridge, USA, 2001.

[4] Y. Weiss. Comparing the mean field method and belief propagation for approximate inference in MRFs. In David Saad and Manfred Opper, editors, *Advanced Mean Field Methods*. MIT Press, 2001.

[5] T. Minka. Divergence measures and message passing. Report 173, Microsoft Research, 2005.

[6] C. Bishop, N. Lawrence, T. Jaakkola, and M. Jordan. Approximating posterior distributions in belief networks using mixtures. In *Advances in Neural Information Processing Systems 10*, 1997.

[7] G. Bouchard and O. Zoeter. Split variational inference. In *Proc. Intl. Conf. Machine Learning*, 2009.

[8] P. Grunwald and A. Dawid. Game theory, maximum entropy, minimum discrepancy, and robust Bayesian decision theory. *Annals of Statistics*, 32(4):1367–1433, 2004.

[9] C. R. Shalizi. Maximum likelihood estimation for q-exponential (tsallis) distributions, 2007. URL http://arxiv.org/abs/math.ST/0701854.

[10] C. Tsallis. Possible generalization of boltzmann-gibbs statistics. *J. Stat. Phys.*, 52:479–487, 1988.

[11] J. Naudts. Deformed exponentials and logarithms in generalized thermostatistics. *Physica A*, 316:323–334, 2002. URL http://arxiv.org/pdf/cond-mat/0203489.

[12] T. D. Sears. *Generalized Maximum Entropy, Convexity, and Machine Learning*. PhD thesis, Australian National University, 2008.

[13] A. Sousa and C. Tsallis. Student's t- and r-distributions: Unified derivation from an entropic variational principle. *Physica A*, 236:52–57, 1994.

[14] C. Tsallis, R. S. Mendes, and A. R. Plastino. The role of constraints within generalized nonextensive statistics. *Physica A: Statistical and Theoretical Physics*, 261:534–554, 1998.

[15] J. Naudts. Generalized thermostatistics based on deformed exponential and logarithmic functions. *Physica A*, 340:32–40, 2004.

[16] J. Naudts. Generalized thermostatistics and mean-field theory. *Physica A*, 332:279–300, 2004.

[17] J. Naudts. Estimators, escort proabilities, and $\phi$-exponential families in statistical physics. *Journal of Inequalities in Pure and Applied Mathematics*, 5(4), 2004.

[18] N. Ding and S. V. N. Vishwanathan. $t$-logistic regression. In Richard Zemel, John Shawe-Taylor, John Lafferty, Chris Williams, and Alan Culota, editors, *Advances in Neural Information Processing Systems 23*, 2010.

[19] A. Rényi. On measures of information and entropy. In *Proc. 4th Berkeley Symposium on Mathematics, Statistics and Probability*, pages 547–561, 1960.

[20] J. D. Lafferty. Additive models, boosting, and inference for generalized divergences. In *Proc. Annual Conf. Computational Learning Theory*, volume 12, pages 125–133. ACM Press, New York, NY, 1999.

[21] I. Csiszár. Information type measures of differences of probability distribution and indirect observations. *Studia Math. Hungarica*, 2:299–318, 1967.

[22] K. Azoury and M. K. Warmuth. Relative loss bounds for on-line density estimation with the exponential family of distributions. *Machine Learning*, 43(3):211–246, 2001. Special issue on Theoretical Advances in On-line Learning, Game Theory and Boosting.

[23] W. Wiegerinck and T. Heskes. Fractional belief propagation. In S. Becker, S. Thrun, and K. Obermayer, editors, *Advances in Neural Information Processing Systems 15*, pages 438–445, 2003.

[24] M. Opper. A Bayesian approach to online learning. In *On-line Learning in Neural Networks*, pages 363–378. Cambridge University Press, 1998.

[25] X. Boyen and D. Koller. Tractable inference for complex stochastic processes. In *UAI*, 1998.

[26] J. S. Rosenthal. *A First Look at Rigorous Probability Theory*. World Scientific Publishing, 2006.

